# Training Stochastic Model Recognition Algorithms as Networks can lead to Maximum Mutual Information Estimation of Parameters

**John S. Bridle**
Royal Signals and Radar Establishment
Great Malvern       Worcs.
UK       WR14 3PS

## ABSTRACT

One of the attractions of neural network approaches to pattern recognition is the use of a discrimination-based training method. We show that once we have modified the output layer of a multi-layer perceptron to provide mathematically correct probability distributions, and replaced the usual squared error criterion with a probability-based score, the result is equivalent to Maximum Mutual Information training, which has been used successfully to improve the performance of hidden Markov models for speech recognition. If the network is specially constructed to perform the recognition computations of a given kind of stochastic model based classifier then we obtain a method for discrimination-based training of the parameters of the models. Examples include an HMM-based word discriminator, which we call an 'Alphanet'.

## 1   INTRODUCTION

It has often been suggested that one of the attractions of an adaptive neural network (NN) approach to pattern recognition is the availability of discrimination-based training (*e.g.* in Multilayer Perceptrons (MLPs) using Back-Propagation). Among the disadvantages of NN approaches are the lack of theory about what can be computed with any particular structure, what can be learned, how to choose a network architecture for a given task, and how to deal with data (such as speech) in which an underlying sequential structure is of the essence. There have been attempts to build internal dynamics into neural networks, using recurrent connections, so that they might deal with sequences and temporal patterns [1, 2], but there is a lack of relevant theory to inform the choice of network type.

Hidden Markov models (HMMs) are the basis of virtually all modern automatic speech recognition systems. They can be seen as an extension of the parametric statistical approach to pattern recognition, to deal (in a simple but principled way) with temporal patterning. Like most parametric models, HMMs are usually trained using within-class maximum-likelihood (ML) methods, and an EM algorithm due to Baum and Welch is particularly attractive (see for instance [3]). However, recently

some success has been demonstrated using discrimination-based training methods, such as the so-called Maximum Mutual Information criterion [4] and Corrective Training[5].

This paper addresses two important questions:

- How can we design Neural Network architectures with at least the desirable properties of methods based on stochastic models (such as hidden Markov models)?

- What is the relationship between the inherently discriminative neural network training and the analogous MMI training of stochastic models?

We address the first question in two steps. Firstly, to make sure that the outputs of our network have the simple mathematical properties of conditional probability distributions over class labels we recommend a generalisation of the logistic nonlinearity; this enables us (but does not require us) to replace the usual squared error criterion with a more appropriate one, based on relative entropy. Secondly, we also have the option of designing networks which exactly implement the recognition computations of a given stochastic model method. (The resulting 'network' may be rather odd, and not very 'neural', but this is engineering, not biology.) As a contribution to the investigation of the second question, we point out that optimising the relative entropy criterion is exactly equivalent to performing Maximum Mutual Information Estimation.

By way of illustration we describe three 'networks' which implement stochastic model classifiers, and show how discrimination training can help.

## 2  TRAINABLE NETWORKS AS PARAMETERISED CONDITIONAL DISTRIBUTION FUNCTIONS

We consider a trainable network, when used for pattern classification, as a vector function $Q(x, \theta)$ from an input vector $x$ to a set of indicators of class membership, $\{Q_j\}$,   $j = 1, \ldots N$. The parameters $\theta$ modify the transfer function. In a multilayer perceptron, for instance, the parameters would be values of weights. Typically, we have a training set of pairs $(x_t, c_t)$,   $t = 1, \ldots T$, of inputs and associated true class labels, and we have to find a value for $\theta$ which specialises the function so that it is consistent with the training set. A common procedure is to minimise $E(\theta)$, the sum of the squares of the differences between the network outputs and true class indicators, or targets:

$$E(\theta) = \sum_{t=1}^{T} \sum_{j=1}^{N} (Q_j(x_t, \theta) - \delta_{j,c_t})^2,$$

where $\delta_{j,c} = 1$ if $j = c$, otherwise 0. $E$ and $Q$ will be written without the $\theta$ argument where the meaning is clear, and we may drop the $t$ subscript.

It is well known that the value of $F(x)$ which minimises the expected value of $(F(x) - y)^2$ is the expected value of $y$ given $x$. The expected value of $\delta_{j,c_t}$ is $P(C = j \mid X = x_t)$, the probability that the class associated with $x_t$ is the $j^{th}$ class.

From now on we shall assume that the desired output of a classifier network is this conditional probability distribution over classes, given the input.

The outputs must satisfy certain simple constraints if they are to be interpretable as a probability distribution. For any input, the outputs must all be positive and they must sum to unity. The use of logistic nonlinearities at the outputs of the network ensures positivity, and also ensures that each output is less than unity. These constraints are appropriate for outputs that are to be interpreted as probabilities of Boolean events, but are not sufficient for 1-from-N classifiers.

Given a set of unconstrained values, $V_j(x)$, we can ensure both conditions by using a Normalised Exponential transformation:

$$Q_j(x) = e^{V_j(x)} / \sum_k e^{V_k(x)}$$

This transformation can be considered a multi-input generalisation of the logistic, operating on the whole output layer. It preserves the rank order of its input values, and is a differentiable generalisation of the 'winner-take-all' operation of picking the maximum value. For this reason we like to refer to it as **softmax**. Like the logistic, it has a simple implementation in transistor circuits [6].

If the network is such that we can be sure the values we have are all positive, it may be more appropriate just to normalise them. In particular, if we can treat them as likelihoods of the data given the possible classes, $L_j(x) = P(X = x \mid C = j)$, then normalisation produces the required conditional distribution (assuming equal prior probabilities for the classes).

## 3 RELATIVE ENTROPY SCORING FOR CLASSIFIERS

In this section we introduce an information-theoretic criterion for training 1-from-N classifier networks, to replace the squared error criterion, both for its intrinsic interest and because of the link to discriminative training of stochastic models. the class with highest likelihood. This is justified by

$$P(c \mid x) = P(x \mid c)P(c)/P(x),$$

if we assume equal priors $P(c)$ (this can be generalised) and see that the denominator $P(x) = \sum_c P(x \mid c)P(c)$ is the same for all classes.

It is also usual to train such classifiers by maximising the data likelihood given the correct classes. Maximum Likelihood (ML) training is appropriate if we are choosing from a family of pdfs which includes the correct one. In most real-life applications of pattern classification we do not have knowledge of the form of the data distributions, although we may have some useful ideas. In that case ML may be a rather bad approach to pdf estimation *for the purpose of pattern classification*, because what matters is the *relative* densities.

An alternative is to optimise a measure of success in pattern classification, and this can make a big difference to performance, particularly when the assumptions about the form of the class pdfs is badly wrong.

To make the likelihoods produced by a SM classifier look like NN outputs we can simply normalise them:

$$Q_j(x) = L_j(x)/\sum_k L_k(x).$$

Then we can use Neural Network optimisation methods to adjust the parameters.

a sum, weighted by the joint probability, of the MI of the joint events

$$I(X,Y) = \sum_{(x,y)} P(X{=}x, Y{=}y) \log \frac{P(X{=}x, Y{=}y)}{P(X{=}x)P(Y{=}y)}$$

For discrimination training of sets of stochastic models, Bahl et.al. suggest maximising the Mutual Information, I, between the training observations and the choice of the corresponding correct class.

$$I(X,C) = \sum_t \log \frac{P(C{=}c_t, X{=}x_t)}{P(C{=}c_t)P(X{=}x)} = \sum_t \log \frac{P(C{=}c_t \mid X{=}x_t)P(X{=}x_t)}{P(C{=}c_t)P(X{=}x)}.$$

$P(C{=}c_t \mid X = x_t)$ should be read as the probability that we choose the correct class for the $t^{\text{th}}$ training example. If we are choosing classes according to the conditional distribution computed using parameters $\theta$ then $P(C = c_t \mid X = x_t) = Q_{c_t}(x, \theta)$, and

$$I(X,C) = \sum_t \log \frac{Q_{c_t}(x_t, \vartheta)}{P(C{=}c_t)} = \sum_t \log Q_{c_t}(x_t, \theta) - \sum_t \log P(C{=}c_t).$$

If the second term involving the priors is fixed, we are left with maximising

$$\sum_t \log Q_{c_t}(x_t, \theta) = -J.$$

The RE-based score we use is $J = -\sum_{t=1}^{T} \sum_{j=1}^{N} P_{jt} \log Q_j(x_t)$, where $P_{jt}$ is the probability of class $j$ associated with input $x_t$ in the training set. If as usual the training set specifies only one true class, $c_t$ for each $x_t$ then $P_{j,t} = \delta_{j,c_t}$ and

$$J = -\sum_{t=1}^{T} \log Q_{c_t}(x_t),$$

the sum of the logs of the outputs for the correct classes.

$J$ can be derived from the Relative Entropy of distribution Q with respect to the true conditional distribution P, averaged over the input distribution:

$$\int dx\, P(X = x)G(Q \mid P), \quad \text{where} \quad G(Q \mid P) = -\sum_c P(c \mid x) \log \frac{P(c \mid x)}{Q_c(x)}.$$

information, cross entropy, asymmetric divergence, directed divergence, I-divergence, and Kullback-Leibler number. RE scoring is the basis for the Boltzmann Machine learning algorithm [7] and has also been proposed and used for adaptive networks with continuous-valued outputs [8, 9, 10, 11], but usually in the form appropriate to separate logistics and independent Boolean targets. An exception is [12].

There is another way of thinking about this 'log-of correct-output' score. Assume that the way we would use the outputs of the network is that, rather than choosing

the class with the largest output, we choose randomly, picking from the distribution specified by the outputs. (Pick class $j$ with probability $Q_j$.) The probability of choosing the class $c_t$ for training sample $\boldsymbol{x}_t$ is simply $Q_{c_t}(\boldsymbol{x}_t)$. The probability of choosing the correct class labels for *all* the training set is $\prod_{t=1}^{T} Q_{c_t}(\boldsymbol{x}_t)$. We simply seek to maximise this probability, or what is equivalent, to minimise minus its log:

$$ J = - \sum_{t=1}^{T} \log Q_{c_t}(\boldsymbol{x}_t). $$

In order to compute the partial derivatives of $J$ wrt to parameters of the network, we first need $\frac{\partial J}{\partial Q_j} = -P_{jt}/Q_j$ The details of the back-propagation depend on the form of the network, but if the final non-linearity is a normalised exponential (softmax),

$$ Q_j(\boldsymbol{x}) = \exp(V_j(\boldsymbol{x})) / \sum_k \exp(V_k(\boldsymbol{x})), \quad \text{then [6]} \quad \frac{\partial J_t}{\partial V_j} = (Q_j(\boldsymbol{x}_t) - \delta_{j,c_t}). $$

We see that the derivative before the output nonlinearity is the difference between the corresponding output and a one-from-N target. We conclude that softmax output stages and 1-from-N RE scoring are natural partners.

# 4  DISCRIMINATIVE TRAINING

In stochastic model (probability-density) based pattern classification we usually compute likelihoods of the data given models for each class, $P(\boldsymbol{x}\,|\,c)$, and choose. So minimising our $J$ criterion is also maximising Bahl's mutual information. (Also see [13].)

# 5  STOCHASTIC MODEL CLASSIFIERS AS NETWORKS
## 5.1  EXAMPLE ONE: A PAIR OF MULTIVARIATE GAUSSIANS

The conditional distribution for a pair of multivariate Gaussian densities with the same arbitrary covariance matrix is a logistic function of a weighted sum of the input coordinates (plus a constant). Therefore, even if we make such incorrect assumptions as equal priors and spherical unit covariances, it is still possible to find values for the parameters of the model (the positions of the means of the assumed distributions) for which the form of the conditional distribution is correct. (The means may be far from the means of the true distributions and from the data means.) Of course in this case we have the alternative of using a weighted-sum logistic unit to compute the conditional probability: the parameters are then the weights.

## 5.2  EXAMPLE TWO: A MULTI-CLASS GAUSSIAN CLASSIFIER

Consider a model in which the distributions for each class are multi-variate Gaussian, with equal isotropic unit variances, and different means, $\{m_j\}$. The probability distribution over class labels, given an observation $\boldsymbol{x}$, is $P(c = j\,|\,\boldsymbol{x}) = e^{V_j} / \sum_k e^{V_k}$, where $V_j = -||\boldsymbol{x} - m_j||^2$. This can be interpreted as a one-layer feed-forward non-linear network. The usual weighted sums are replaced by squared Euclidean distances, and the usual logistic output non-linearities are replaced by a normalised exponential.

For a particular two-dimensional 10-class problem, derived from Peterson and Barney's formant data, we have demonstrated [6] that training such a network can cause the $m$s to move from their "natural" positions at the data means (the in-class maximum likelihood estimates), and this can improve classification performance on unseen data (from 68% correct to 78%).

## 5.3   EXAMPLE THREE: ALPHANETS

Consider a set of hidden Markov models (HMMs), one for each word, each parameterised by a set of state transition probabilities, $\{a_{ij}^k\}$, and observation likelihood functions $\{b_j^k(x)\}$, where $a_{ij}^k$ is the probability that in model $k$ state $i$ will be followed by state $j$, and $b_j^k(x)$ is the likelihood of model $k$ emitting observation $x$ from state $j$. For simplicity we insist that the end of the word pattern corresponds to state $N$ of a model.

The likelihood, $L_k(x_1^M)$ of model $k$ generating a given sequence $x_1^M \triangleq x_1, \ldots, x_M$ is a sum, over all sequences of states, of the joint likelihood of that state sequence and the data:

$$L_k(x_1^M) = \sum_{s_1 \ldots s_M} \prod_{t=2}^M a_{s_{t-1}, s_t}^k \, b_{s_t}^k(x_t) \quad \text{with} \quad s_M = N.$$

This can be computed efficiently via the forward recursion [3]

$$\alpha_{jk}(t) = b_j^k(x_t) \sum_i a_{ij}^k \alpha_{ik}(t-1), \qquad \text{giving} \qquad L_k(x_1^M) = \alpha_{Nk}(M),$$

which we can think of as a recurrent network. (Note that $t$ is used as a time index here.)

If the observation sequence $x_1^M$ could only have come from one of a set of known, equally likely models, then the posterior probability that it was from model $k$ is

$$P(C{=}k \mid x_1^M) = Q_k(x_1^M) = L_k(x_1^M) \Big/ \sum_l L_l(x_1^M).$$

These numbers are the output of our special "recurrent neural network" for isolated word discrimination, which we call an "Alphanet" [14]. Backpropagation of partial derivatives of the $J$ score has the form of the **backward** recurrence used in the Baum-Welch algorithm, but they include discriminative terms, and we obtain the gradient of the relative entropy/mutual information.

## 6   CONCLUSIONS

Discrimination-based training is different from within-class parameter estimation, and it may be useful. (Also see [15].) Discrimination-based training for stochastic models and for networks are not distinct, and in some cases can be mathematically identical.

The notion of specially constructed 'network' architectures which implement stochastic model recognition algorithms provides a way to construct fertile hybrids. For instance, a Gaussian classifier (or a HMM classifier) can be preceeded by a nonlinear transformation (perhaps based on semilinear logistics) and all the parameters

of the system adjusted together. This seems a useful approach to automating the discovery of 'feature detectors'.

# References

[1] R P Lippmann. Review of neural networks for speech recognition. *Neural Computation*, 1(1), 1989.

[2] R L Watrous. Connectionist speech recognition using the temporal flow model. In *Proc. IEEE Workshop on Speech Recognition*, June 1988.

[3] A B Poritz. Hidden Markov models: a guided tour. In *Proc. IEEE Int. Conf. Acoustics Speech and Signal Processing*, pages 7–13, 1988.

[4] L R Bahl, P F Brown, P V de Souza, and R L Mercer. Maximum mutual information estimation of hidden Markov model parameters. In *Proc. IEEE Int. Conf. Acoustics Speech and Signal Processing*, pages 49–52, 1986.

[5] L R Bahl, P F Brown, P V de Souza, and R L Mercer. A new algorithm for the estimation of HMM parameters. In *Proc. IEEE Int. Conf. Acoustics Speech and Signal Processing*, pages 493–496, 1988.

[6] J S Bridle. Probabilistic interpretation of feedforward classification network outputs, with relationships to statistical pattern recognition. In F Fougelman-Soulie and J Hérault, editors, *Neuro-computing: algorithms, architectures and applications*, Springer-Verlag, 1989.

[7] D H Ackley, G E Hinton, and T J Sejnowski. A learning algorithm for Boltzmann machines. *Cognitive Science*, 9:147–168, 1985.

[8] L Gillick. Probability scores for backpropagation networks. July 1987. Personal communication.

[9] G E Hinton. *Connectionist Learning Procedures*. Technical Report CMU-CS-87-115, Carnegie Mellon University Computer Science Department, June 1987.

[10] E B Baum and F Wilczek. Supervised learning of probability distributions by neural networks. In D Anderson, editor, *Neural Information Processing Systems*, pages 52–61, Am. Inst. of Physics, 1988.

[11] S Solla, E Levin, and M Fleisher. Accelerated learning in layered neural networks. *Complex Systems*, January 1989.

[12] E Yair and A Gersho. The Boltzmann Perceptron Network: a soft classifier. In D Touretzky, editor, *Advances in Neural Information Processing Systems 1*, San Mateo, CA: Morgan Kaufmann, 1989.

[13] P S Gopalakrishnan, D Kanevsky, A Nadas, D Nahamoo, and M A Picheny. Decoder selection based on cross-entropies. In *Proc. IEEE Int. Conf. Acoustics Speech and Signal Processing*, pages 20–23, 1988.

[14] J S Bridle. Alphanets: a recurrent 'neural' network architecture with a hidden Markov model interpretation. *Speech Communication*, Special Neurospeech issue, February 1990.

[15] L Niles, H Silverman, G Tajchman, and M Bush. How limited training data can allow a neural network to out-perform an 'optimal' classifier. In *Proc. IEEE Int. Conf. Acoustics Speech and Signal Processing*, 1989.
